# Gaussian-process factor analysis for low-dimensional single-trial analysis of neural population activity

**Byron M. Yu**[1,2,4]**, John P. Cunningham**[1]**, Gopal Santhanam**[1]**,**
**Stephen I. Ryu**[1,3]**, Krishna V. Shenoy**[1,2]
[1]Department of Electrical Engineering, [2]Neurosciences Program,
[3]Department of Neurosurgery, Stanford University, Stanford, CA 94305
{byronyu,jcunnin,gopals,seoulman,shenoy}@stanford.edu

**Maneesh Sahani**[4]
[4]Gatsby Computational Neuroscience Unit, UCL
London, WC1N 3AR, UK
maneesh@gatsby.ucl.ac.uk

## Abstract

We consider the problem of extracting smooth, low-dimensional *neural trajectories* that summarize the activity recorded simultaneously from tens to hundreds of neurons on individual experimental trials. Current methods for extracting neural trajectories involve a two-stage process: the data are first "denoised" by smoothing over time, then a static dimensionality reduction technique is applied. We first describe extensions of the two-stage methods that allow the degree of smoothing to be chosen in a principled way, and account for spiking variability that may vary both across neurons and across time. We then present a novel method for extracting neural trajectories, Gaussian-process factor analysis (GPFA), which unifies the smoothing and dimensionality reduction operations in a common probabilistic framework. We applied these methods to the activity of 61 neurons recorded simultaneously in macaque premotor and motor cortices during reach planning and execution. By adopting a goodness-of-fit metric that measures how well the activity of each neuron can be predicted by all other recorded neurons, we found that GPFA provided a better characterization of the population activity than the two-stage methods.

## 1  Introduction

Neural responses are typically studied by averaging noisy spiking activity across multiple experimental trials to obtain firing rates that vary smoothly over time. However, particularly in cognitive tasks (such as motor planning or decision making) where the neural responses are more a reflection of internal processing rather than external stimulus drive, the timecourse of the neural responses may differ on nominally identical trials. In such settings, it is critical that the neural data not be averaged across trials, but instead be analyzed on a trial-by-trial basis [1, 2, 3, 4].

Single-trial analyses can leverage the simultaneous monitoring of large populations of neurons *in vivo*, currently ranging from tens to hundreds in awake, behaving animals. The approach adopted by recent studies is to consider each neuron being recorded as a noisy sensor reflecting the time-evolution of an underlying neural process [3, 5, 6, 7, 8, 9, 10]. The goal is to uncover this neural process by extracting a smooth, low-dimensional *neural trajectory* from the noisy, high-dimensional recorded activity on a single-trial basis. The neural trajectory provides a compact representation of

the high-dimensional recorded activity as it evolves over time, thereby facilitating data visualization and studies of neural dynamics under different experimental conditions.

A common method to extract neural trajectories is to first estimate a smooth firing rate profile for each neuron on a single trial (e.g., by convolving each spike train with a Gaussian kernel), then apply a static dimensionality reduction technique (e.g., principal components analysis, PCA) [8, 11]. Smooth firing rate profiles may also be obtained by averaging across a small number of trials (if the neural timecourses are believed to be similar on different trials) [6, 7, 9, 10], or by applying more advanced statistical methods for estimating firing rate profiles from single spike trains [12, 13]. Numerous linear and non-linear dimensionality reduction techniques exist, but to our knowledge only PCA [8, 9, 11] and locally linear embedding (LLE) [6, 7, 10, 14] have been applied in this context to neural data.

While this two-stage method of performing smoothing then dimensionality reduction has provided informative low-dimensional views of neural population activity, there are several aspects that can be improved. (i) For kernel smoothing, the degree of smoothness is often chosen in an *ad hoc* way. We would instead like to learn the appropriate degree of smoothness from the data. Because the operations of kernel smoothing, PCA, and LLE are all non-probabilistic, standard likelihood techniques for model selection are not applicable. Even if a probabilistic dimensionality reduction algorithm is used, the likelihoods would not be comparable because different smoothing kernels yield different smoothed data. (ii) The same kernel width is typically used for all spike trains, which implicitly assumes that the neural population activity evolves with a single timescale. We would instead like to allow for the possibility that the system operates under multiple timescales. (iii) PCA and LLE have no explicit noise model and, therefore, have difficulty distinguishing between spiking noise (whose variance may vary both across neurons and across time) and changes in the underlying low-dimensional neural state. (iv) Because the smoothing and dimensionality reduction are performed sequentially, there is no way for the dimensionality reduction algorithm to influence the degree or form of smoothing used. This is relevant both to the identification of the low-dimensional space, as well as to the extraction of single-trial neural trajectories.

We first briefly describe relatively straightforward extensions of the two-stage methods that can help to address issues (i) and (iii) above. For (i), we adopt a goodness-of-fit metric that measures how well the activity of each neuron can be predicted by the activity of all other recorded neurons, based on data not used for model fitting. This metric can be used to compare different smoothing kernels and allows for the degree of smoothness to be chosen in a principled way. In Section 6, we will use this as a common metric by which different methods for extracting neural trajectories are compared. For (iii), we can apply the square-root transform to stabilize the spiking noise variance and factor analysis (FA) [15] to explicitly model possibly different independent noise variances for different neurons. These extensions are detailed in Sections 2 and 3.

Next, we introduce Gaussian-process factor analysis (GPFA), which unifies the smoothing and dimensionality reduction operations in a common probabilistic framework. GPFA takes steps toward addressing all of the issues (i)–(iv) described above, and is shown in Section 6 to provide a better characterization of the recorded population activity than the two-stage methods. Because GPFA performs the smoothing and dimensionality reduction operations simultaneously rather than sequentially, the degree of smoothness and the relationship between the low-dimensional neural trajectory and the high-dimensional recorded activity can be jointly optimized. Different dimensions in the low-dimensional space (within which the neural state evolves) can have different timescales, whose optimal values can be found automatically by fitting the GPFA model to the recorded activity. As in FA, GPFA specifies an explicit noise model that allows different neurons to have different independent noise variances. The time series model involves Gaussian processes (GP), which only require the specification of the correlation structure of the neural state over time.

A critical assumption when attempting to extract a low-dimensional neural trajectory is that the recorded activity evolves within a low-dimensional manifold. Previous studies have typically assumed that the neural trajectories lie in a three-dimensional space for ease of visualization. In this work, we will investigate whether this low-dimensional assumption is justified in the context of motor preparation and execution and, if so, attempt to identify the appropriate dimensionality. Sections 2 and 3 detail GPFA and the goodness-of-fit metric, respectively. Section 4 relates GPFA to dynamical systems approaches. After describing the experimental setup in Section 5, we apply the

developed methods to neural activity recorded in premotor and motor cortices during reach planning and execution in Section 6.

## 2 Gaussian-process factor analysis

The motivation for GPFA can be traced back to the use of PCA for extracting informative low-dimensional views of high-dimensional neural data. Consider spike counts taken in non-overlapping time bins. PCA (or its probabilistic form, PPCA [15]) attempts to find the directions in the high-dimensional data with greatest variance. This is problematic for neural data for two reasons. First, because neurons with higher mean counts are known to exhibit higher count variances, the directions found by PCA tend to be dominated by the most active neurons. Second, PCA assumes that the spiking noise variance is time independent; however, neurons are known to change their firing rates, and therefore noise variances, over time. A possible solution is to replace the Gaussian likelihood model of PPCA with a point-process [5] or Poisson [3] likelihood model. Here, we consider a simpler approach that preserves computational tractability. The square-root transform is known to both stabilize the variance of Poisson counts and allow Poisson counts to be more closely modeled by a Gaussian distribution, especially at low Poisson means [16]. Thus, the two issues above can be largely resolved by applying PCA/PPCA to square-rooted spike counts, rather than raw spike counts. However, the spiking noise can deviate from a Poisson distribution [17], in which case the noise variance is not entirely stabilized. As will be shown in Section 6, the square-rooted counts can be better characterized by further replacing PCA/PPCA with FA [15], which allows different neurons to have different independent noise variances.

In this work, we extend FA for use with time series data. PCA, PPCA, and FA are all static dimensionality reduction techniques. In other words, none of them take into account time labels when applied to time series data; the time series data are simply treated as a collection of data points. GPFA is an extension of FA that can leverage the time label information to provide more powerful dimensionality reduction. The GPFA model is simply a set of factor analyzers (one per timepoint, each with identical parameters) that are linked together in the low-dimensional state space by a Gaussian process (GP) [18] prior. Introducing the GP allows for the specification of a correlation structure across the low-dimensional states at different timepoints. For example, if the system underlying the time series data is believed to evolve smoothly over time, we can specify that the system's state should be more similar between nearby timepoints than between faraway timepoints. Extracting a smooth, low-dimensional neural trajectory can therefore be viewed as a compromise between the low-dimensional projection of each data point found by FA and the desire to string them together using a smooth function over time. The GPFA model can also be obtained by letting time indices play the role of inputs in the semiparametric latent factor model [19].

The following is a mathematical description of GPFA. Let $\mathbf{y}_{:,t} \in \mathbb{R}^{q \times 1}$ be the high-dimensional vector of square-rooted spike counts recorded at timepoint $t \in \{1, \ldots, T\}$, where $q$ is the number of neurons being recorded simultaneously. We seek to extract a corresponding low-dimensional latent *neural state* $\mathbf{x}_{:,t} \in \mathbb{R}^{p \times 1}$ at each timepoint, where $p$ is the dimensionality of the state space ($p < q$). For notational convenience, we group the neural states from all timepoints into a *neural trajectory* denoted by the matrix $X = [\mathbf{x}_{:,1}, \ldots, \mathbf{x}_{:,T}] \in \mathbb{R}^{p \times T}$. Similarly, the observations can be grouped into a matrix $Y = [\mathbf{y}_{:,1}, \ldots, \mathbf{y}_{:,T}] \in \mathbb{R}^{q \times T}$. We define a linear-Gaussian relationship between the observations $\mathbf{y}_{:,t}$ and neural states $\mathbf{x}_{:,t}$

$$\mathbf{y}_{:,t} \mid \mathbf{x}_{:,t} \sim \mathcal{N} \left( C\mathbf{x}_{:,t} + \mathbf{d}, \ R \right), \tag{1}$$

where $C \in \mathbb{R}^{q \times p}$, $\mathbf{d} \in \mathbb{R}^{q \times 1}$, and $R \in \mathbb{R}^{q \times q}$ are model parameters to be learned. As in FA, we constrain the covariance matrix $R$ to be diagonal, where the diagonal elements are the independent noise variances of each neuron. In general, different neurons can have different independent noise variances. Although a Gaussian is not strictly a distribution on square-rooted counts, its use in (1) preserves computational tractability.

The neural states $\mathbf{x}_{:,t}$ at different timepoints are related through Gaussian processes, which embody the notion that the neural trajectories should be smooth. We define a separate GP for each dimension of the state space indexed by $i \in \{1, \ldots, p\}$

$$\mathbf{x}_{i,:} \sim \mathcal{N} \left( \mathbf{0}, \ K_i \right), \tag{2}$$

where $\mathbf{x}_{i,:} \in \mathbb{R}^{1 \times T}$ is the $i$th row of $X$ and $K_i \in \mathbb{R}^{T \times T}$ is the covariance matrix for the $i$th GP [20]. The form of the GP covariance can be chosen to provide different smoothing properties on the neural trajectories. In this work, we chose the commonly-used squared exponential (SE) covariance function

$$K_i(t_1, t_2) = \sigma_{f,i}^2 \cdot \exp\left(-\frac{(t_1 - t_2)^2}{2 \cdot \tau_i^2}\right) + \sigma_{n,i}^2 \cdot \delta_{t_1,t_2}, \tag{3}$$

where $K_i(t_1, t_2)$ denotes the $(t_1, t_2)$th entry of $K_i$ and $t_1, t_2 \in \{1, \ldots, T\}$. The SE covariance is defined by its signal variance $\sigma_{f,i}^2 \in \mathbb{R}_+$, characteristic timescale $\tau_i \in \mathbb{R}_+$, and noise variance $\sigma_{n,i}^2 \in \mathbb{R}_+$. Due to redundancy in the scale of $X$ and $C$, we fix the scale of $X$ and allow $C$ to be learned unconstrained, without loss of generality. By direct analogy to FA, we defined the prior distribution of the neural state $\mathbf{x}_{:,t}$ at each timepoint $t$ to be $\mathcal{N}(\mathbf{0}, I)$ by setting $\sigma_{f,i}^2 = 1 - \sigma_{n,i}^2$, where $0 < \sigma_{n,i}^2 \leq 1$. Furthermore, because we seek to extract smooth neural trajectories, we set $\sigma_{n,i}^2$ to a small value ($10^{-3}$). Thus, the timescale $\tau_i$ is the only (hyper)parameter of the SE covariance that is learned. The SE is an example of a stationary covariance; other stationary and non-stationary GP covariances [18] can be applied in a seamless way.

The parameters of the GPFA model can be learned in a straightforward way using the expectation-maximization (EM) algorithm. In the E-step, the Gaussian posterior distribution $P(X \mid Y)$ can be computed exactly because the $\mathbf{x}_{:,t}$ and $\mathbf{y}_{:,t}$ across all timepoints are jointly Gaussian, by definition. In the M-step, the parameters updates for $C$, $\mathbf{d}$, and $R$ can be expressed in closed form. The characteristic timescales $\tau_i$ can be updated using any gradient optimization technique. Note that the degree of smoothness (defined by the timescales) and the relationship between the low-dimensional neural trajectory and the high-dimensional recorded activity (defined by $C$) are jointly optimized. Furthermore, a different timescale is learned for each state dimension indexed by $i$. For the results shown in Section 6, the parameters $C$, $\mathbf{d}$, and $R$ were initialized using FA, and the $\tau_i$ were initialized to 100 ms. Although the learned timescales were initialization-dependent, their distributions were similar for different initializations. In particular, most learned timescales were less than 150 ms, but there were usually one or two larger timescales around 300 and 500 ms.

Once the GPFA model is learned, we can apply a post-processing step to orthonormalize the columns of $C$. Applying the singular value decomposition, $C\mathbf{x}_{:,t}$ can be rewritten as $U_C(D_C V_C' \mathbf{x}_{:,t})$, where the columns of $U_C \in \mathbb{R}^{q \times p}$ are orthonormal and $\tilde{\mathbf{x}}_{:,t} = D_C V_C' \mathbf{x}_{:,t} \in \mathbb{R}^{p \times 1}$ is referred to as the *orthonormalized neural state* at timepoint $t$. While each dimension of $\mathbf{x}_{:,t}$ possesses a single characteristic timescale, each dimension of $\tilde{\mathbf{x}}_{:,t}$ represents a mixture of timescales defined by the columns of $V_C$. An advantage of considering $\tilde{\mathbf{x}}_{:,t}$ rather than $\mathbf{x}_{:,t}$ is that the elements of $\tilde{\mathbf{x}}_{:,t}$ (and the corresponding columns of $U_C$) are ordered by the amount of data covariance explained. In contrast, the elements of $\mathbf{x}_{:,t}$ (and the corresponding columns of $C$) have no particular order. Especially when the number of state dimensions $p$ is large, the ordering facilitates the identification and visualization of the dimensions of the orthonormalized neural trajectory that are most important for explaining the recorded activity. Because the columns of $U_C$ are orthonormal, one can readily picture how the low-dimensional trajectory relates to the high-dimensional space of recorded activity, in much the same spirit as for PCA. This orthonormalization procedure is also applicable to PPCA and FA. In fact, it is through this orthonormalization procedure that the principal directions found by PPCA are equated to those found by PCA.

## 3  Leave-neuron-out prediction error

We would like to directly compare GPFA to the two-stage methods described in Section 1. Neither the classic approach of comparing cross-validated likelihoods nor the Bayesian approach of comparing marginal likelihoods is applicable here, for the same reason that they cannot be used to select the appropriate degree of smoothness in the two-stage methods. Namely, when the data are altered by different pre-smoothing operations (or the lack thereof in the case of GPFA), the likelihoods are no longer comparable. Instead, we adopted the goodness-of-fit metric mentioned in Section 1, whereby a prediction error is computed based on trials not used for model fitting. The idea is to leave out one neuron at a time and ask how well each method is able to predict the activity of that neuron, given the activity of all other recorded neurons. For GPFA, the model prediction for neuron $j$ is $\hat{\mathbf{y}}_{j,:} = E[\mathbf{y}_{j,:} \mid Y_{-j,:}]$, where $\mathbf{y}_{j,:}$ is the $j$th row of $Y$ and $Y_{-j,:} \in \mathbb{R}^{(q-1) \times T}$ represents all but

the $j$th row of $Y$. The model prediction can be computed analytically because all variables in $Y$ are jointly Gaussian, by definition. Model predictions using PPCA and FA are analogous, but each timepoint is considered individually. The prediction error is defined as the sum-of-squared errors between the model prediction and the observed square-rooted spike count across all neurons and timepoints.

One way to compute the GPFA model prediction is via the low-dimensional state space. One can first estimate the neural trajectory using all but the $j$th neuron $P\left(X \mid Y_{-j,:}\right)$, then map this estimate back out into the space of recorded activity for the $j$th neuron using (1) to obtain $\hat{\mathbf{y}}_{j,:}$. Equivalently, one can convert $P\left(X \mid Y_{-j,:}\right)$ into its orthonormalized form before mapping it out into the space of recorded activity using the $j$th row of $U_C$. Because the orthonormalized dimensions are ordered, we can evaluate the prediction error using only the top $\tilde{p}$ orthonormalized dimensions of $\tilde{\mathbf{x}}_{:,t}$, where $\tilde{p} \in \{1, \ldots, p\}$. This reduced GPFA model can make use of a larger number $p$ of timescales than its effective dimensionality $\tilde{p}$.

## 4  Linear and non-linear dynamical systems

Another way to extract neural trajectories is by defining a parametric dynamical model that describes how the low-dimensional neural state evolves over time. A first-order linear auto-regressive (AR) model [5] captures linear Markovian dynamics. Such a model can be expressed as a Gaussian process, since the state variables are jointly Gaussian. This can be shown by defining a separate first-order AR model for each state dimension indexed by $i \in \{1, \ldots, p\}$

$$x_{i,t+1} \mid x_{i,t} \sim \mathcal{N}\left(a_i x_{i,t},\ \sigma_i^2\right). \tag{4}$$

Given enough time ($t \rightarrow \infty$) and $|a_i| < 1$, the model will settle into a stationary state that is equivalent to (2) with

$$K_i(t_1, t_2) = \frac{\sigma_i^2}{1 - a_i^2}\, a_i^{|t_1 - t_2|}, \tag{5}$$

as in [21]. Different covariance structures $K_i$ can be obtained by going from a first-order to an $n$th-order AR model. One drawback of this approach is that it is usually not easy to construct an $n$th-order AR model with a specified covariance structure. In contrast, the GP approach described in Section 2 requires only the specification of the covariance structure, thus allowing different smoothing properties to be applied in a seamless way. AR models are generally less computationally demanding than those based on GP, but this advantage shrinks as the order of the AR model grows. Another difference is that (5) does not contain an independent noise term $\sigma_{n,i}^2 \cdot \delta_{t_1,t_2}$ as in (3). The innovations noise $\sigma_i^2$ in (4) is involved in setting the smoothness of the time series, as shown in (5). Thus, (4) would need to be augmented to explicitly capture departures from the AR model.

One may also consider defining a non-linear dynamical model [3], which typically has a richer set of dynamical behaviors than linear models. The identification of the model parameters provides insight into the dynamical rules governing the time-evolution of the system under study. However, especially in exploratory data analyses, it may be unclear what form this model should take. Even if an appropriate non-linear model can be identified, learning such a model can be unstable and slow due to approximations required [3]. In contrast, learning the GPFA model is stable and approximation-free, as described in Section 2. The use of GPFA can be viewed as a practical way of going beyond a first-order linear AR model without having to commit to a particular non-linear system, while retaining computational tractability.

## 5  Behavioral task and neural recordings

The details of the neural recordings and behavioral task can be found elsewhere [22]. Briefly, a rhesus macaque performed delayed center-out reaches to visual targets presented on a fronto-parallel screen. On a given trial, the peripheral reach target was presented at one of 14 possible locations – two distances (60 and 100 mm) and seven directions (0, 45, 90, 135, 180, 225, 315°). Delay periods were randomly chosen between 200 and 700 ms. Neural activity was recorded using a 96-electrode array (Cyberkinetics, Foxborough, MA) in dorsal premotor and motor cortices. Only those units (61 single and multi-units, experiment `G20040123`) with robust delay period activity were included in our analyses.

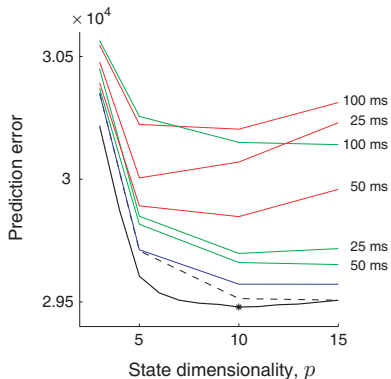

Figure 1: Prediction errors of two-stage methods (PPCA: red, FA: green), first-order AR model (blue), GPFA (dashed black), and reduced GPFA (solid black), computed using 4-fold cross-validation. Labels at right are standard deviations of Gaussian kernels (referred to as *kernel widths*) for the two-stage methods. For reduced GPFA, the horizontal axis corresponds to $\tilde{p}$ rather than $p$, where the prediction error is computed using only the top $\tilde{p}$ orthonormalized dimensions of a GPFA model fit with $p = 15$. Star indicates minimum of solid black curve. Analyses in this figure are based on 56 trials for the reach target at distance 60 mm and direction 135°.

## 6 Results

We considered neural data for one reach target at a time, ranging from 200 ms before reach target onset to movement end. This period comprised the 200 ms pre-target time, the randomly chosen delay period (200–700 ms), the monkey's reaction time (mean±s.d.: 293±48 ms), and the duration of the monkey's reach (269±40 ms). Spike counts were taken in non-overlapping 20 ms bins, then square-rooted. For the two-stage methods, these square-rooted counts were smoothed over time using a Gaussian kernel. We also considered smoothing spike trains directly, which yielded qualitatively similar results for the two-stage methods.

Using the goodness-of-fit metric described in Section 3, we can find the appropriate degree of smoothness for the two-stage methods. Fig. 1 shows the prediction error for PPCA (red) and FA (green) for different kernel widths and state dimensionalities. There are two primary findings. First, FA yielded lower prediction error than PPCA across a range of kernel widths and state dimensionalities. The reason is that FA allows different neurons to have different independent noise variances. Second, for these data, the optimal smoothing kernel width (s.d. of Gaussian kernel) is approximately 40 ms for both FA and PPCA. This was found using a denser sweep of the kernel width than shown in Fig. 1.

It is tempting to try to relate this optimal smoothing kernel width (40 ms) to the timescales $\tau_i$ learned by GPFA, since the SE covariance has the same shape as the Gaussian smoothing kernel. However, nearly all of the timescales learned by GPFA are greater than 40 ms. This apparent mismatch can be understood by considering the *equivalent kernel* of the SE covariance [23], which takes on a sinc-like shape whose main lobe is generally far narrower than a Gaussian kernel with the same width parameter. It is therefore reasonable that the timescales learned by GPFA are larger than the optimal smoothing kernel width.

The same goodness-of-fit metric can be used to compare the two-stage methods, parametric dynamical models, and GPFA. The parametric dynamical model considered in this work is a first-order AR model described by (2) and (5), coupled with the linear-Gaussian observation model (1). Note that a separate stationary, one-dimensional first-order AR model is defined for each of the $p$ latent dimensions. As shown in Fig. 1, the first-order AR model (blue) yielded lower prediction error than the two-stage methods (PPCA: red, FA: green). Furthermore, GPFA (dashed black) performed as well or better than the two-stage methods and the first-order AR model, regardless of the state dimensionality or kernel width used. As described in Section 3, the prediction error can also be computed for a reduced GPFA model (solid black) using only the top $\tilde{p}$ orthonormalized dimensions, in this case based on a GPFA model fit with $p = 15$ state dimensions. By definition, the dashed and solid black lines coincide at $\tilde{p} = 15$. The solid black curve reaches its minimum at $\tilde{p} = 10$ (referred to as $p^*$). Thus, removing the lowest five orthonormalized dimensions decreased the GPFA prediction error. Furthermore, this prediction error was lower than when fitting the GPFA model directly with $p = 10$ (dashed black).

These latter findings can be understood by examining the orthonormalized neural trajectories extracted by GPFA shown in Fig. 2. The traces plotted are the orthonormalized form of $E[X \mid Y]$. The panels are arranged in decreasing order of data covariance explained. The top orthonormalized dimensions indicate fluctuations in the recorded population activity shortly after target onset (red

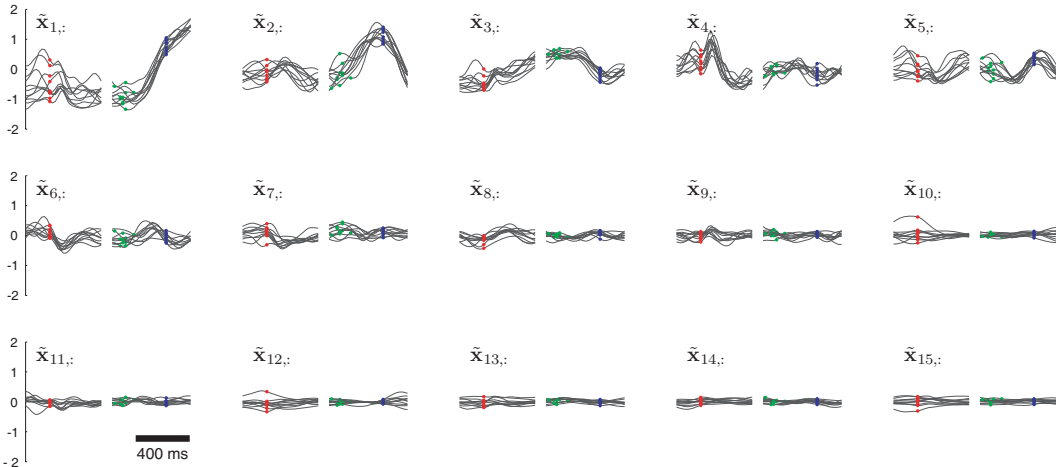

Figure 2: Orthonormalized neural trajectories for GPFA with $p = 15$. Each panel corresponds to one of the 15 dimensions of the orthonormalized neural state, which is plotted versus time. The orthonormalized neural trajectory for one trial comprises one black trace from each panel. Dots indicate time of reach target onset (red), go cue (green), and movement onset (blue). Due to differing trial lengths, the traces on the left/right half of each panel are aligned on target/movement onset for clarity. However, the GPFA model was fit using entire trials with no gaps. Note that the polarity of these traces is arbitrary, as long as it is consistent with the polarity of $U_C$. Each trajectory corresponds to planning and executing a reach to the target at distance 60 mm and direction 135°. For clarity, only 10 trials with delay periods longer than 400 ms are plotted.

dots) and again after the go cue (green dots). Furthermore, the neural trajectories around the time of the arm movement are well-aligned on movement onset. These observations are consistent with previous analyses of the same dataset [22], as well as other studies of neural activity collected during similar tasks in the same cortical areas. Whereas the top 10 orthonormalized dimensions (upper and middle rows) show repeatable temporal structure across trials, the bottom five dimensions (lower row) appear to be largely capturing noise. These "noise dimensions" could be limiting GPFA's predictive power. This is confirmed by Fig. 1: when the bottom five orthonormalized dimensions were removed, the GPFA prediction error decreased.

It still remains to be explained why the GPFA prediction error using only the top 10 orthonormalized dimensions is lower than that obtained by directly fitting a GPFA model with $p = 10$. Each panel in Fig. 2 represents a mixture of 15 characteristic timescales. Thus, the top 10 orthonormalized dimensions can make use of up to 15 timescales. However, a GPFA model fit with $p = 10$ can have at most 10 timescales. By fitting a GPFA model with a large number of state dimensions $p$ (each with its own timescale) and taking only the top $\tilde{p} = p^*$ orthonormalized dimensions, we can obtain neural trajectories whose effective dimensionality is smaller than the number of timescales at play.

Based on the solid black line in Fig. 1 and Fig. 2, we consider the effective dimensionality of the recorded population activity to be $p^* = 10$. In other words, the linear subspace within which the recorded activity evolved during reach planning and execution for this particular target was 10-dimensional. Across the 14 reach targets, the effective dimensionality ranged from 8 to 12. All major trends seen in Fig. 1 were preserved across all reach targets.

## 7   Conclusion

GPFA offers a flexible and intuitive framework for extracting neural trajectories, whose learning algorithm is stable, approximation-free, and simple to implement. Because only the GP covariance structure needs to be specified, GPFA is particularly attractive for exploratory data analyses, where the rules governing the dynamics of the system under study are unknown. Based on the trajectories obtained by GPFA, one can then attempt to define an appropriate dynamical model that describes how the neural state evolves over time.

Compared with two-stage methods, the choice of GP covariance allows for more explicit specification of the smoothing properties of the low-dimensional trajectories. This is important when investigating (possibly subtle) properties of the system dynamics. For example, one may wish to ask whether the system exhibits second-order dynamics by examining the extracted trajectories. In this case, it is critical that second-order effects not be built-in by the smoothness assumptions used to extract the trajectories. With GPFA, it is possible to select a triangular GP covariance that assumes smoothness in position, but not in velocity. In contrast, it is unclear how to choose the shape of the smoothing kernel to achieve this in the two-stage methods.

In future work, we would like to couple the covariance structure of the one-dimensional GPs, which would allow for a richer description of the multi-dimensional neural state $\mathbf{x}_{:,t}$ evolving over time. We also plan to apply non-stationary GP kernels, since the neural data collected during a behavioral task are usually non-stationary. In addition, we would like to extend GPFA by allowing for the discovery of non-linear manifolds and applying point-process likelihood models.

## Acknowledgments

This work was supported by NIH-NINDS-CRCNS 5-R01-NS054283-03, NSF, NDSEGF, Gatsby, SGF, CDRF, BWF, ONR, Sloan, and Whitaker. We would like to thank Dr. Mark Churchland, Melissa Howard, Sandra Eisensee, and Drew Haven.

## References

[1] K. L. Briggman, H. D. I. Abarbanel, and W. B. Kristan Jr. *Science*, 307(5711):896–901, Feb. 2005.

[2] K. L. Briggman, H. D. I. Abarbanel, and W. B. Kristan Jr. *Curr Opin Neurobiol*, 16(2):135–144, 2006.

[3] B. M. Yu, A. Afshar, G. Santhanam, S. I. Ryu, K. V. Shenoy, and M. Sahani. In Y. Weiss, B. Scholkopf, and J. Platt, eds., *Adv Neural Info Processing Sys 18*, pp. 1545–1552. MIT Press, 2006.

[4] M. M. Churchland, B. M. Yu, M. Sahani, and K. V. Shenoy. *Curr Opin Neurobiol*, 17(5):609–618, 2007.

[5] A. C. Smith and E. N. Brown. *Neural Comput*, 15(5):965–991, 2003.

[6] M. Stopfer, V. Jayaraman, and G. Laurent. *Neuron*, 39:991–1004, Sept. 2003.

[7] S. L. Brown, J. Joseph, and M. Stopfer. *Nat Neurosci*, 8(11):1568–1576, Nov. 2005.

[8] R. Levi, R. Varona, Y. I. Arshavsky, M. I. Rabinovich, and A. I. Selverston. *J Neurosci*, 25(42):9807–9815, Oct. 2005.

[9] O. Mazor and G. Laurent. *Neuron*, 48:661–673, Nov. 2005.

[10] B. M. Broome, V. Jayaraman, and G. Laurent. *Neuron*, 51:467–482, Aug. 2006.

[11] M. A. L. Nicolelis, L. A. Baccala, R. C. S. Lin, and J. K. Chapin. *Science*, 268(5215):1353–1358, 1995.

[12] I. DiMatteo, C. R. Genovese, and R. E. Kass. *Biometrika*, 88(4):1055–1071, 2001.

[13] J. P. Cunningham, B. M. Yu, K. V. Shenoy, and M. Sahani. In J. Platt, D. Koller, Y. Singer, and S. Roweis, eds., *Adv Neural Info Processing Sys 20*. MIT Press, 2008.

[14] S. T. Roweis and L. K. Saul. *Science*, 290(5500):2323–2326, Dec. 2000.

[15] S. Roweis and Z. Ghahramani. *Neural Comput*, 11(2):305–345, 1999.

[16] N. A. Thacker and P. A. Bromiley. The effects of a square root transform on a Poisson distributed quantity. Technical Report 2001-010, University of Manchester, 2001.

[17] D. J. Tolhurst, J. A. Movshon, and A. F. Dean. *Vision Res*, 23(8):775–785, 1983.

[18] C. E. Rasmussen and C. K. I. Williams. *Gaussian processes for machine learning*. MIT Press, 2006.

[19] Y. W. Teh, M. Seeger, and M. I. Jordan. In R. G. Cowell and Z. Ghahramani, eds., *Proceedings of the Tenth International Workshop on Artificial Intelligence and Statistics (AISTATS)*. Society for Artificial Intelligence and Statistics, 2005.

[20] N. D. Lawrence and A. J. Moore. In Z. Ghahramani, ed., *Proceedings of the 24th Annual International Conference on Machine Learning (ICML 2007)*, pp. 481–488. Omnipress, 2007.

[21] R. E. Turner and M. Sahani. *Neural Comput*, 19(4):1022–1038, 2007.

[22] M. M. Churchland, B. M. Yu, S. I. Ryu, G. Santhanam, and K. V. Shenoy. *J Neurosci*, 26(14):3697–3712, Apr. 2006.

[23] P. Sollich and C. K. I. Williams. In L. K. Saul, Y. Weiss, and L. Bottou, eds., *Advances in Neural Information Processing Systems 17*, pp. 1313–1320. MIT Press, 2005.

